# Generalization Errors and Learning Curves for Regression with Multi-task Gaussian Processes

**Kian Ming A. Chai**
School of Informatics, University of Edinburgh,
10 Crichton Street, Edinburgh EH8 9AB, UK
`k.m.a.chai@ed.ac.uk`

## Abstract

We provide some insights into how task correlations in multi-task Gaussian process (GP) regression affect the generalization error and the learning curve. We analyze the asymmetric two-tasks case, where a secondary task is to help the learning of a primary task. Within this setting, we give bounds on the generalization error and the learning curve of the primary task. Our approach admits intuitive understandings of the multi-task GP by relating it to single-task GPs. For the case of one-dimensional input-space under optimal sampling with data only for the secondary task, the limitations of multi-task GP can be quantified explicitly.

## 1 Introduction

Gaussian processes (GPs) (see e.g., [1]) have been applied to many practical problems. In recent years, a number of models for multi-task learning with GPs have been proposed to allow different tasks to leverage on one another [2–5]. While it is generally assumed that learning multiple tasks together is beneficial, we are not aware of any work that quantifies such benefits, other than PAC-based theoretical analysis for multi-task learning [6–8]. Following the tradition of the theoretical works on GPs in machine learning, our goal is to quantify the benefits using average-case analysis.

We concentrate on the asymmetric two-tasks case, where the secondary task is to help the learning of the primary task. Within this setting, the main parameters are (1) the degree of "relatedness" $\rho$ between the two tasks, and (2) the ratio $\pi_S$ of total training data for the secondary task. While higher $|\rho|$ and lower $\pi_S$ is clearly more beneficial to the primary task, the extent and manner that this is so has not been clear. To address this, we measure the benefits using generalization error, learning curve and optimal error, and investigate the influence of $\rho$ and $\pi_S$ on these quantities.

We will give non-trivial lower and upper bounds on the generalization error and the learning curve. Both types of bounds are important in providing assurance on the quality of predictions: an upper bound provides an estimate of the amount of training data needed to attain a minimum performance level, while a lower bound provides an understanding of the limitations of the model [9]. Our approach relates multi-task GPs to single-task GPs and admits intuitive understandings of multi-task GPs. For one-dimensional input-space under optimal sampling with data only for the secondary task, we show the limit to which error for the primary task can be reduced. This dispels any misconception that abundant data for the secondary task can remedy no data for the primary task.

## 2 Preliminaries and problem statement

### 2.1 Multi-task GP regression model and setup

The multi-task Gaussian process regression model in [5] learns $M$ related functions $\{f_m\}_{m=1}^M$ by placing a zero mean GP prior which directly induces correlations between tasks. Let $y_m$ be an

observation of the $m$th function at $\boldsymbol{x}$. Then the model is given by

$$\langle f_m(\boldsymbol{x})f_{m'}(\boldsymbol{x}')\rangle \stackrel{\text{def}}{=} K^{\text{f}}_{mm'}k^{\text{x}}(\boldsymbol{x},\boldsymbol{x}') \qquad y_m \sim \mathcal{N}(f_m(\boldsymbol{x}),\sigma_m^2), \tag{1}$$

where $k^{\text{x}}$ is a covariance function over inputs, and $K^{\text{f}}$ is a positive semi-definite matrix of inter-task similarities, and $\sigma_m^2$ is the noise variance for the $m$th task.

The current focus is on the two tasks case, where the secondary task $S$ is to help improve the performance of the primary task $T$; this is the *asymmetric multi-task learning* as coined in [10]. We fix $K^{\text{f}}$ to be a correlation matrix, and let the variance be explained fully by $k^{\text{x}}$ (the converse has been done in [5]). Thus $K^{\text{f}}$ is fully specified by the correlation $\rho \in [-1,1]$ between the two tasks. We further fix the noise variances of the two tasks to be the same, say $\sigma_{\text{n}}^2$. For the training data, there are $n_T$ (resp. $n_S$) observations at locations $X_T$ (resp. $X_S$) for task $T$ (resp. $S$). We use $n \stackrel{\text{def}}{=} n_T + n_S$ for the total number of observations, $\pi_S \stackrel{\text{def}}{=} n_S/n$ for the proportion of observations for task $S$, and also $X \stackrel{\text{def}}{=} X_T \cup X_S$. The aim is to infer the noise-free response $f_{T*}$ for task $T$ at $\boldsymbol{x}_*$. See Figure 1.

The covariance matrix of the noisy training data is $K(\rho) + \sigma_{\text{n}}^2 I$, where

$$K(\rho) \stackrel{\text{def}}{=} \begin{pmatrix} K^{\text{x}}_{TT} & \rho K^{\text{x}}_{TS} \\ \rho K^{\text{x}}_{ST} & K^{\text{x}}_{SS} \end{pmatrix}; \tag{2}$$

and $K^{\text{x}}_{TT}$ (resp. $K^{\text{x}}_{SS}$) is the matrix of covariances (due to $k^{\text{x}}$) between locations in $X_T$ (resp. $X_S$); $K^{\text{x}}_{TS}$ is the matrix of cross-covariances from locations in $X_T$ to locations in $X_S$; and $K^{\text{x}}_{ST}$ is $K^{\text{x}}_{TS}$ transposed. The posterior variance at $\boldsymbol{x}_*$ for task $T$ is

$$\sigma_T^2(\boldsymbol{x}_*,\rho,\sigma_{\text{n}}^2,X_T,X_S) = k_{**} - \boldsymbol{k}_*^{\text{T}}(K(\rho)+\sigma_{\text{n}}^2 I)^{-1}\boldsymbol{k}_*, \quad \text{where } \boldsymbol{k}_*^{\text{T}} \stackrel{\text{def}}{=} \left((\boldsymbol{k}^{\text{x}}_{T*})^{\text{T}}\ \rho(\boldsymbol{k}^{\text{x}}_{S*})^{\text{T}}\right); \tag{3}$$

and $k_{**}$ is the prior variance at $\boldsymbol{x}_*$, and $\boldsymbol{k}^{\text{x}}_{T*}$ (resp. $\boldsymbol{k}^{\text{x}}_{S*}$) is the vector of covariances (due to $k^{\text{x}}$) between locations in $X_T$ (resp. $X_S$) and $\boldsymbol{x}_*$. Where appropriate and clear from context, we will suppress some of the parameters in $\sigma_T^2(\boldsymbol{x}_*,\rho,\sigma_{\text{n}}^2,X_T,X_S)$, or use $X$ for $(X_T,X_S)$. Note that $\sigma_T^2(\rho) = \sigma_T^2(-\rho)$, so that $\sigma_T^2(1)$ is the same as $\sigma_T^2(-1)$; for brevity, we only write the former.

If the GP prior is correctly specified, then the posterior variance (3) is also the generalization error at $\boldsymbol{x}_*$ [1, §7.3]. The latter is defined as $\langle (f_T^\star(\boldsymbol{x}_*) - \bar{f}_T(\boldsymbol{x}_*))^2 \rangle_{f_T^\star}$, where $\bar{f}_T(\boldsymbol{x}_*)$ is the posterior mean at $\boldsymbol{x}_*$ for task $T$, and the expectation is taken over the distribution from which the true function $f_T^\star$ is drawn. In this paper, in order to distinguish succinctly from the generalization error introduced in the next section, we use posterior variance to mean the generalization error at $\boldsymbol{x}_*$. Note that the actual $y$-values observed at $X$ do not effect the posterior variance at any test location.

**Problem statement**  Given the above setting, the aim is to investigate how training observations for task $S$ can benefit the predictions for task $T$. We measure the benefits using generalization error, learning curve and optimal error, and investigate how these quantities vary with $\rho$ and $\pi_S$.

### 2.2  Generalization errors, learning curves and optimal errors

We outline the general approach to obtain the generalization error and the learning curve [1, §7.3] under our setting, where we have two tasks and are concerned with the primary task $T$. Let $p(\boldsymbol{x})$ be the probability density, common to both tasks, from which test and training locations are drawn, and assume that the GP prior is correctly specified. The generalization error for task $T$ is obtained by averaging the posterior variance for task $T$ over $\boldsymbol{x}_*$, and the learning curve for task $T$ is obtained by averaging the generalization error over training sets $X$:

generalization error: $\qquad \epsilon_T(\rho,\sigma_{\text{n}}^2,X_T,X_S) \stackrel{\text{def}}{=} \int \sigma_T^2(\boldsymbol{x}_*,\rho,\sigma_{\text{n}}^2,X_T,X_S)p(\boldsymbol{x}_*)\mathrm{d}\boldsymbol{x}_*$ (4)

learning curve: $\qquad \epsilon_T^{\text{avg}}(\rho,\sigma_{\text{n}}^2,\pi_S,n) \stackrel{\text{def}}{=} \int \epsilon_T(\rho,\sigma_{\text{n}}^2,X_T,X_S)p(X)\mathrm{d}X,$ (5)

where the training locations in $X$ are drawn i.i.d, that is, $p(X)$ factorizes completely into a product of $p(\boldsymbol{x})$s. Besides averaging $\epsilon_T$ to obtain the learning curve, one may also use the optimal experimental design methodology and minimize $\epsilon_T$ over $X$ to find the optimal generalization error [11, chap. II]:

optimal error: $\qquad \epsilon_T^{\text{opt}}(\rho,\sigma_{\text{n}}^2,\pi_S,n) \stackrel{\text{def}}{=} \min_X \epsilon_T(\rho,\sigma_{\text{n}}^2,X_T,X_S).$ (6)

Both $\epsilon_T(0,\sigma_{\text{n}}^2,X_T,X_S)$ and $\epsilon_T(1,\sigma_{\text{n}}^2,X_T,X_S)$ reduce to single-task GP cases; the former discards training observations at $X_S$, while the latter includes them. Similar analogues to single-task GP cases for $\epsilon_T^{\text{avg}}(0,\sigma_{\text{n}}^2,\pi_S,n)$ and $\epsilon_T^{\text{avg}}(1,\sigma_{\text{n}}^2,\pi_S,n)$, and $\epsilon_T^{\text{opt}}(0,\sigma_{\text{n}}^2,\pi_S,n)$ and $\epsilon_T^{\text{opt}}(1,\sigma_{\text{n}}^2,\pi_S,n)$ can be obtained. Note that $\epsilon_T^{\text{avg}}$ and $\epsilon_T^{\text{opt}}$ are well-defined since $\pi_S n = n_S \in \mathbb{N}_0$ by the definition of $\pi_S$.

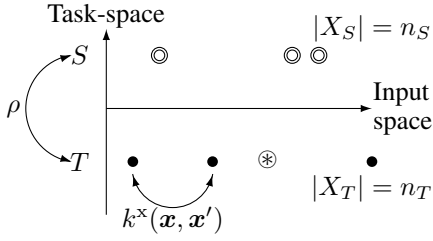

Figure 1: The two tasks $S$ and $T$ have task correlation $\rho$. The data set $X_T$ (resp. $X_S$) for task $T$ (resp. $S$) consists of the •s (resp. ⊚s). The test location $\boldsymbol{x}_*$ for task $T$ is denoted by ⊛.

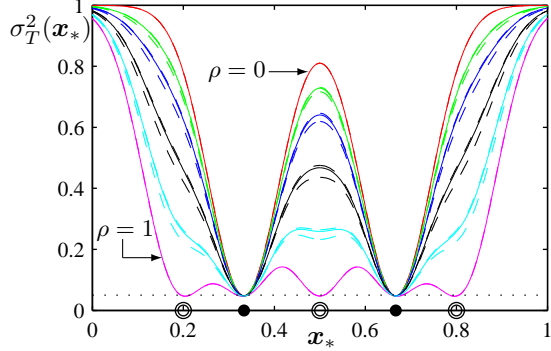

Figure 2: The posterior variances of each test location within $[0, 1]$ given data •s at $1/3$ and $2/3$ for task $T$, and ⊚s at $1/5$, $1/2$ and $4/5$ for task $S$.

### 2.3 Eigen-analysis

We now state known results of eigen-analysis used in this paper. Let $\bar{\kappa} \stackrel{\text{def}}{=} \kappa_1 > \kappa_2 > \dots$ and $\phi_1(\cdot), \phi_2(\cdot), \dots$ be the eigenvalues and eigenfunctions of the covariance function $k^{\text{x}}$ under the measure $p(\boldsymbol{x})\mathrm{d}\boldsymbol{x}$: they satisfy the integral equation $\int k^{\text{x}}(\boldsymbol{x}, \boldsymbol{x}')\phi_i(\boldsymbol{x})p(\boldsymbol{x})\mathrm{d}\boldsymbol{x} = \kappa_i\phi_i(\boldsymbol{x}')$. Let $\bar{\lambda} \stackrel{\text{def}}{=} \lambda_1 > \lambda_2 > \dots > \lambda_{n_S} \stackrel{\text{def}}{=} \underline{\lambda}$ be the eigenvalues of $K^{\text{x}}_{SS}$. If the locations in $X_S$ are sampled from $p(\boldsymbol{x})$, then $\kappa_i = \lim_{n_S \to \infty} \lambda_i/n_S$, $i = 1 \dots n_S$; see e.g., [1, §4.3.2] and [12, Theorem 3.4]. However, for finite $n_S$ used in practice, the estimate $\lambda_i/n_S$ for $\kappa_i$ is better for the larger eigenvalues than for the smaller ones. Additionally, in one-dimension with uniform $p(\boldsymbol{x})$ on the unit interval, if $k^{\text{x}}$ satisfies the Sacks-Ylvisaker conditions of order $r$, then $\kappa_i \propto (\pi i)^{-2r-2}$ in the limit $i \to \infty$ [11, Proposition IV.10, Remark IV.2]. Broadly speaking, an order $r$ process is exactly $r$ times mean square differentiable. For example, the stationary Ornstein-Uhlenbeck process is of order $r = 0$.

## 3 Generalization error

In this section, we derive expressions for the generalization error (and the bounds thereon) for the two-tasks case in terms of the single-task one. To illustrate and further motivate the problem, Figure 2 plots the posterior variance $\sigma_T^2(\boldsymbol{x}_*, \rho)$ as a function of $\boldsymbol{x}_*$ given two observations for task $T$ and three observations for task $S$. We roughly follow [13, Fig. 2], and use squared exponential covariance function with length-scale 0.11 and noise variance $\sigma_{\text{n}}^2 = 0.05$. Six solid curves are plotted, corresponding, from top to bottom, to $\rho^2 = 0$, $1/8$, $1/4$, $1/2$, $3/4$ and $1$. The two dashed curves enveloping each solid curve are the lower and upper bounds derived in this section; the dashed curves are hardly visible because the bounds are rather tight. The dotted line is the prior noise variance.

Similar to the case of single-task learning, each training point creates a depression on the $\sigma_T^2(\boldsymbol{x}_*, \rho)$ surface [9, 13]. However, while each training point for task $T$ creates a "full" depression that reaches the prior noise variance (horizontal dotted line at 0.05), the depression created by each training point for task $S$ depends on $\rho$, "deeper" depressions for larger $\rho^2$. From the figure, and also from definition, it is clear that the following trivial bounds on $\sigma_T^2(\boldsymbol{x}_*, \rho)$ hold:

**Proposition 1.** *For all $\boldsymbol{x}_*$, $\sigma_T^2(\boldsymbol{x}_*, 1) \leqslant \sigma_T^2(\boldsymbol{x}_*, \rho) \leqslant \sigma_T^2(\boldsymbol{x}_*, 0)$.*

Integrating wrt to $\boldsymbol{x}_*$ then gives the following corollary:

**Corollary 2.** *$\epsilon_T(1, \sigma_{\text{n}}^2, X_T, X_S) \leqslant \epsilon_T(\rho, \sigma_{\text{n}}^2, X_T, X_S) \leqslant \epsilon_T(0, \sigma_{\text{n}}^2, X_T, X_S)$.*

Sections 3.2 and 3.3 derive lower and upper bounds that are tighter than the above trivial bounds. Prior to the bounds, we consider a degenerate case to illustrate the limitations of multi-task learning.

### 3.1 The degenerate case of no training data for primary task

It is clear that if there is no training data for the secondary task, that is, if $X_S = \emptyset$, then $\sigma_T^2(\boldsymbol{x}_*1) = \sigma_T^2(\boldsymbol{x}_*, \rho) = \sigma_T^2(\boldsymbol{x}_*0)$ for all $\boldsymbol{x}_*$ and $\rho$. In the converse case where there is no training data for the primary task, that is, $X_T = \emptyset$, we instead have the following proposition:

**Proposition 3.** *For all $\boldsymbol{x}_*$, $\sigma_T^2(\boldsymbol{x}_*, \rho, \emptyset, X_S) = \rho^2 \sigma_T^2(\boldsymbol{x}_*, 1, \emptyset, X_S) + (1 - \rho^2)k_{**}$.*

*Proof.*
$$\sigma_T^2(\boldsymbol{x}_*, \rho, \emptyset, X_S) = k_{**} - \rho^2 (\boldsymbol{k}_{S*}^{\mathrm{x}})^{\mathrm{T}} (K_{SS}^{\mathrm{x}} + \sigma_{\mathrm{n}}^2 I)^{-1} \boldsymbol{k}_{S*}^{\mathrm{x}}$$
$$= (1 - \rho^2)k_{**} + \rho^2 \left[ k_{**} - (\boldsymbol{k}_{S*}^{\mathrm{x}})^{\mathrm{T}} (K_{SS}^{\mathrm{x}} + \sigma_{\mathrm{n}}^2 I)^{-1} \boldsymbol{k}_{S*}^{\mathrm{x}} \right]$$
$$= (1 - \rho^2)k_{**} + \rho^2 \sigma_T^2(\boldsymbol{x}_*, 1, \emptyset, X_S). \qquad \square$$

Hence the posterior variance is a weighted average of the prior variance $k_{**}$ and the posterior variance at perfect correlation. When the cardinality of $X_S$ increases under infill asymptotics [14, §3.3],

$$\lim_{n_S \to \infty} \sigma_T^2(\boldsymbol{x}_*, 1, \emptyset, X_S) = 0 \quad \implies \quad \lim_{n_S \to \infty} \sigma_T^2(\boldsymbol{x}_*, \rho, \emptyset, X_S) = (1 - \rho^2)k_{**}. \quad (7)$$

This is the limit for the posterior variance at *any* test location for task $T$, if one has training data only for the secondary task $S$. This is because a correlation of $\rho$ between the tasks prevents any training location for task $S$ from having correlation higher than $\rho$ with a test location for task $T$. Suppose correlations in the input-space are given by an isotropic covariance function $k^{\mathrm{x}}(|\boldsymbol{x} - \boldsymbol{x}'|)$. If we translate correlations into distances between data locations, then any training location from task $S$ is beyond a certain radius from any test location for task $T$. In contrast, a training location from task $T$ may lay arbitrarily close to a test location for task $T$, subject to the constraints of noise.

We obtain the generalization error in this degenerate case, by integrating Proposition 3 wrt $p(\boldsymbol{x}_*)\mathrm{d}\boldsymbol{x}_*$ and using the fact that the mean prior variance is given by the sum of the process eigenvalues.

**Corollary 4.** $\epsilon_T(\rho, \sigma_{\mathrm{n}}^2, \emptyset, X_S) = \rho^2 \epsilon_T(1, \sigma_{\mathrm{n}}^2, \emptyset, X_S) + (1 - \rho^2) \sum_{i=1}^{\infty} \kappa_i$.

## 3.2 A lower bound

When $X_T \neq \emptyset$, the correlations between locations in $X_T$ and locations in $X_S$ complicate the situation. However, since $\sigma_T^2(\rho)$ is a continuous and monotonically decreasing function of $\rho$, there exists an $\alpha \in [0, 1]$, which depends on $\rho$, $\boldsymbol{x}_*$ and $X$, such that $\sigma_T^2(\rho) = \alpha \sigma_T^2(1) + (1 - \alpha)\sigma_T^2(0)$. That $\alpha$ depends on $\boldsymbol{x}_*$ obstructs further analysis. The next proposition gives a lower bound $\underline{\sigma}_T^2(\rho)$ of the same form satisfying $\sigma_T^2(1) \leqslant \underline{\sigma}_T^2(\rho) \leqslant \sigma_T^2(\rho)$, where the mixing proportion is independent of $\boldsymbol{x}_*$.

**Proposition 5.** *Let $\underline{\sigma}_T^2(\boldsymbol{x}_*, \rho) \stackrel{\mathrm{def}}{=} \rho^2 \sigma_T^2(\boldsymbol{x}_*, 1) + (1 - \rho^2)\sigma_T^2(\boldsymbol{x}_*, 0)$. Then for all $\boldsymbol{x}_*$:*

*(a)* $\underline{\sigma}_T^2(\boldsymbol{x}_*, \rho) \leqslant \sigma_T^2(\boldsymbol{x}_*, \rho)$

*(b)* $\sigma_T^2(\boldsymbol{x}_*, \rho) - \underline{\sigma}_T^2(\boldsymbol{x}_*, \rho) \leqslant \rho^2(\sigma_T^2(\boldsymbol{x}_*, 0) - \sigma_T^2(\boldsymbol{x}_*, 1))$

*(c)* $\arg\max_{\rho^2} \left[ \sigma_T^2(\boldsymbol{x}_*, \rho) - \underline{\sigma}_T^2(\boldsymbol{x}_*, \rho) \right] \geqslant 1/2$.

The proofs are in supplementary material §S.2. The lower bound $\underline{\sigma}_T^2(\rho)$ depends explicitly on $\rho^2$. It depends implicitly on $\pi_S$, which is the proportion of observations for task $S$, through the gap between $\sigma_T^2(1)$ and $\sigma_T^2(0)$. If there is no training data for the primary task, i.e., if $\pi_S = 1$, the bound reduces to Proposition 3, and becomes exact for all values of $\rho$. If $\pi_S = 0$, the bound is also exact. For $\pi_S \notin \{0, 1\}$, the bound is exact when $\rho \in \{-1, 0, 1\}$. As from Figure 2 and later from our simulation results in section 5.3, this bound is rather tight. Part (b) of the proposition states the tightness of the bound: it is no more than factor $\rho^2$ of the gap between the trivial bounds $\sigma_T^2(0)$ and $\sigma_T^2(1)$. Part (c) of the proposition says that the bound is least tight for a value of $\rho^2$ greater than $1/2$.

We provide an intuition on Proposition 5a. Let $\bar{f}_1$ (resp. $\bar{f}_0$) be the posterior mean of the single-task GP when $\rho = 1$ (resp. $\rho = 0$). Contrasted with the multi-task predictor $\bar{f}_T$, $\bar{f}_1$ directly involves the noisy observations for task $T$ at $X_S$, so it has more information on task $T$. Hence, predicting $\bar{f}_1(\boldsymbol{x}_*)$ gives the trivial lower bound $\sigma_T^2(1)$ on $\sigma_T^2(\rho)$. The tighter bound $\underline{\sigma}_T^2(\rho)$ is obtained by "throwing away" information and predicting $\bar{f}_1(\boldsymbol{x}_*)$ with probability $\rho^2$ and $\bar{f}_0(\boldsymbol{x}_*)$ with probability $(1 - \rho^2)$.

Finally, the next corollary is readily obtained from Proposition 5a by integrating wrt $p(\boldsymbol{x}_*)\mathrm{d}\boldsymbol{x}_*$. This is possible because $\rho$ is independent of $\boldsymbol{x}_*$.

**Corollary 6.** *Let $\underline{\epsilon}_T(\rho, \sigma_{\mathrm{n}}^2, X_T, X_S) \stackrel{\mathrm{def}}{=} \rho^2 \epsilon_T(1, \sigma_{\mathrm{n}}^2, X_T, X_S) + (1 - \rho^2)\epsilon_T(0, \sigma_{\mathrm{n}}^2, X_T, X_S)$. Then $\underline{\epsilon}_T(\rho, \sigma_{\mathrm{n}}^2, X_T, X_S) \leqslant \epsilon_T(\rho, \sigma_{\mathrm{n}}^2, X_T, X_S)$.*

## 3.3 An upper bound via equivalent isotropic noise at $X_S$

The following question motivates our upper bound: if the training locations in $X_S$ had been observed for task $T$ rather than for task $S$, *what is the variance $\tilde{\sigma}_{\mathrm{n}}^2$ of the equivalent isotropic noise at $X_S$ so*

that the posterior variance remains the same? To answer this question, we first refine the definition of $\sigma_T^2(\cdot)$ to include a different noise variance parameter $s^2$ for the $X_S$ observations:

$$\sigma_T^2(\boldsymbol{x}_*, \rho, \sigma_\mathrm{n}^2, s^2, X_T, X_S) \stackrel{\text{def}}{=} k_{**} - \boldsymbol{k}_*^\mathrm{T} \left[ K(\rho) + \begin{pmatrix} \sigma_\mathrm{n}^2 I & 0 \\ 0 & s^2 I \end{pmatrix} \right]^{-1} \boldsymbol{k}_*; \tag{8}$$

cf. (3). We may suppress the parameters $\boldsymbol{x}_*$, $X_T$ and $X_S$ when writing $\sigma_T^2(\cdot)$. The variance $\tilde{\sigma}_\mathrm{n}^2$ of the equivalent isotropic noise is a function of $\boldsymbol{x}_*$ defined by the equation

$$\sigma_T^2(\boldsymbol{x}_*, 1, \sigma_\mathrm{n}^2, \tilde{\sigma}_\mathrm{n}^2) = \sigma_T^2(\boldsymbol{x}_* \rho, \sigma_\mathrm{n}^2, \sigma_\mathrm{n}^2). \tag{9}$$

For any $\boldsymbol{x}_*$ there is always a $\tilde{\sigma}_\mathrm{n}^2$ that satisfies the equation because the difference

$$\Delta(\rho, \sigma_\mathrm{n}^2, s^2) \stackrel{\text{def}}{=} \sigma_T^2(\rho, \sigma_\mathrm{n}^2, \sigma_\mathrm{n}^2) - \sigma_T^2(1, \sigma_\mathrm{n}^2, s^2) \tag{10}$$

is a continuous and monotonically decreasing function of $s^2$. To make progress, we seek an upper bound $\bar{\sigma}_\mathrm{n}^2$ for $\tilde{\sigma}_\mathrm{n}^2$ that is independent of the choice of $\boldsymbol{x}_*$: $\Delta(\rho, \sigma_\mathrm{n}^2, \bar{\sigma}_\mathrm{n}^2) \leqslant 0$ for all test locations. Of interest is the tight upper bound $\bar{\bar{\sigma}}_\mathrm{n}^2$, which is the minimum possible $\bar{\sigma}_\mathrm{n}^2$, given in the next proposition.

**Proposition 7.** *Let $\bar{\lambda}$ be the maximum eigenvalue of $K_{SS}^\mathrm{x}$, $\beta \stackrel{\text{def}}{=} \rho^{-2} - 1$ and $\bar{\bar{\sigma}}_\mathrm{n}^2 \stackrel{\text{def}}{=} \beta(\bar{\lambda} + \sigma_\mathrm{n}^2) + \sigma_\mathrm{n}^2$. Then for all $\boldsymbol{x}_*$, $\sigma_T^2(\boldsymbol{x}_*, \rho, \sigma_\mathrm{n}^2, \sigma_\mathrm{n}^2) \leqslant \sigma_T^2(\boldsymbol{x}_*, 1, \sigma_\mathrm{n}^2, \bar{\bar{\sigma}}_\mathrm{n}^2)$. The bound is tight in this sense: for any $\bar{\sigma}_\mathrm{n}^2$, if $\forall \boldsymbol{x}_* \, \sigma_T^2(\boldsymbol{x}_*, \rho, \sigma_\mathrm{n}^2, \sigma_\mathrm{n}^2) \leqslant \sigma_T^2(\boldsymbol{x}_*, 1, \sigma_\mathrm{n}^2, \bar{\sigma}_\mathrm{n}^2)$, then $\forall \boldsymbol{x}_* \, \sigma_T^2(\boldsymbol{x}_*, \rho, \sigma_\mathrm{n}^2, \bar{\sigma}_\mathrm{n}^2) \leqslant \sigma_T^2(\boldsymbol{x}_*, 1, \sigma_\mathrm{n}^2, \bar{\bar{\sigma}}_\mathrm{n}^2)$.*

*Proof sketch.* Matrix $K(\rho)$ may be factorized as

$$K(\rho) = \begin{pmatrix} I & 0 \\ 0 & \rho I \end{pmatrix} \begin{pmatrix} K_{TT}^\mathrm{x} & K_{TS}^\mathrm{x} \\ K_{ST}^\mathrm{x} & \rho^{-2} K_{SS}^\mathrm{x} \end{pmatrix} \begin{pmatrix} I & 0 \\ 0 & \rho I \end{pmatrix}. \tag{11}$$

By using this factorization in the posterior variance (8) and taking out the $\begin{pmatrix} I & 0 \\ 0 & \rho I \end{pmatrix}$ factors, we obtain

$$\sigma_T^2(\rho, \sigma_\mathrm{n}^2, s^2) = k_{**} - (\boldsymbol{k}_*^\mathrm{x})^\mathrm{T} [\Sigma(\rho, \sigma_\mathrm{n}^2, s^2)]^{-1} \boldsymbol{k}_*^\mathrm{x}, \tag{12}$$

where $(\boldsymbol{k}_*^\mathrm{x})^\mathrm{T} \stackrel{\text{def}}{=} ((\boldsymbol{k}_{T*}^\mathrm{x})^\mathrm{T}, (\boldsymbol{k}_{S*}^\mathrm{x})^\mathrm{T})$ and

$$\Sigma(\rho, \sigma_\mathrm{n}^2, s^2) \stackrel{\text{def}}{=} \begin{pmatrix} K_{TT}^\mathrm{x} & K_{TS}^\mathrm{x} \\ K_{ST}^\mathrm{x} & \rho^{-2} K_{SS}^\mathrm{x} \end{pmatrix} + \begin{pmatrix} \sigma_\mathrm{n}^2 I & 0 \\ 0 & \rho^{-2} s^2 I \end{pmatrix} = \Sigma(1, \sigma_\mathrm{n}^2, s^2) + \beta \begin{pmatrix} 0 & 0 \\ 0 & K_{SS}^\mathrm{x} + s^2 I \end{pmatrix}.$$

The second expression for $\Sigma$ makes clear that, in the terms of $\sigma_T^2(\rho, \sigma_\mathrm{n}^2, \sigma_\mathrm{n}^2)$, having data $X_S$ for task $S$ is equivalent to an additional correlated noise at these observations for task $T$. This expression motivates the question that began this section. Note that $\rho^{-2} \geqslant 1$, and hence $\beta \geqslant 0$.

The increase in posterior variance due to having $X_S$ at task $S$ with noise variance $\sigma_\mathrm{n}^2$ rather than having them at task $T$ with noise variance $s^2$ is given by $\Delta(\rho, \sigma_\mathrm{n}^2, s^2)$, which we may now write as

$$\Delta(\rho, \sigma_\mathrm{n}^2, s^2) = (\boldsymbol{k}_*^\mathrm{x})^\mathrm{T} \left[ (\Sigma(1, \sigma_\mathrm{n}^2, s^2))^{-1} - (\Sigma(\rho, \sigma_\mathrm{n}^2, \sigma_\mathrm{n}^2))^{-1} \right] \boldsymbol{k}_*^\mathrm{x}. \tag{13}$$

Recall that we seek an upper bound $\bar{\sigma}_\mathrm{n}^2$ for $\tilde{\sigma}_\mathrm{n}^2$ such that $\Delta(\rho, \sigma_\mathrm{n}^2, \bar{\sigma}_\mathrm{n}^2) \leqslant 0$ for all test locations. In general, this requires $\bar{\bar{\sigma}}_\mathrm{n}^2 \stackrel{\text{def}}{=} \beta(\bar{\lambda} + \sigma_\mathrm{n}^2) + \sigma_\mathrm{n}^2 \leqslant \bar{\sigma}_\mathrm{n}^2$; details can be found in supplementary material §S.3. The tightness $\bar{\bar{\sigma}}_\mathrm{n}^2$ is evident from the construction. $\qquad \square$

Intuitively, $\sigma_T^2(\boldsymbol{x}_*, 1, \sigma_\mathrm{n}^2, \bar{\bar{\sigma}}_\mathrm{n}^2)$ is the tight upper bound because it inflates the noise (co)variance at $X_S$ just sufficiently, from $(\beta K_{SS}^\mathrm{x} + \sigma_\mathrm{n}^2 I / \rho^2)$ to $\bar{\bar{\sigma}}_\mathrm{n}^2 I$. Analogously, the tight lower bound on $\tilde{\sigma}_\mathrm{n}^2$ is given by $\underline{\underline{\sigma}}_\mathrm{n}^2 \stackrel{\text{def}}{=} \beta(\underline{\lambda} + \sigma_\mathrm{n}^2) + \sigma_\mathrm{n}^2$. In summary, $\rho^{-2} \sigma_\mathrm{n}^2 \leqslant \underline{\underline{\sigma}}_\mathrm{n}^2 \leqslant \tilde{\sigma}_\mathrm{n}^2 \leqslant \bar{\bar{\sigma}}_\mathrm{n}^2 \leqslant \bar{\sigma}_\mathrm{n}^2$, where the first inequality is obtained by substituting in zero for $\underline{\lambda}$ in $\underline{\underline{\sigma}}_\mathrm{n}^2$. Hence observing $X_S$ at $S$ is at most as "noisy" as an additional $\beta(\bar{\lambda} + \sigma_\mathrm{n}^2)$ noise variance, and at least as "noisy" as an additional $\beta(\underline{\lambda} + \sigma_\mathrm{n}^2)$ noise variance. Since $\beta$ decreases with $|\rho|$, the additional noise variances are smaller when $|\rho|$ is larger, i.e., when the task $S$ is more correlated with task $T$.

We give a description of how the above bounds scale with $n_S$, using the results stated in section 2.3. For large enough $n_S$, we may write $\bar{\lambda} \approx n_S \bar{\kappa}$ and $\underline{\lambda} \approx n_S \kappa_{n_S}$. Furthermore, for uniformly distributed inputs in the one-dimension unit interval, if the covariance function satisfies Sacks-Ylvisaker conditions of order $r$, then $\kappa_{n_S} = \Theta\left((\pi n_S)^{-2r-2}\right)$, so that $\underline{\lambda} = \Theta\left((\pi n_S)^{-2r-1}\right)$. Since $\bar{\bar{\sigma}}_\mathrm{n}^2$ and $\underline{\underline{\sigma}}_\mathrm{n}^2$ are linear in $\bar{\lambda}$ and $\underline{\lambda}$, we have $\bar{\bar{\sigma}}_\mathrm{n}^2 = \rho^{-2} \sigma_\mathrm{n}^2 + \beta \, \Theta(n_S)$ and $\underline{\underline{\sigma}}_\mathrm{n}^2 = \rho^{-2} \sigma_\mathrm{n}^2 + \beta \, \Theta\left(n_S^{-2r-1}\right)$. For the upper bound $\bar{\bar{\sigma}}_\mathrm{n}^2$, note that although it scales linearly with $n_S$, the eigenvalues of $K(1)$ scales with $n$, thus $\sigma_T^2(1, \sigma_\mathrm{n}^2, \bar{\bar{\sigma}}_\mathrm{n}^2)$ depends on $\pi_S \stackrel{\text{def}}{=} n_S / n$. In contrast the lower bound $\underline{\underline{\sigma}}_\mathrm{n}^2$ is dominated by $\rho^{-2} \sigma_\mathrm{n}^2$, so that $\sigma_T^2(1, \sigma_\mathrm{n}^2, \underline{\underline{\sigma}}_\mathrm{n}^2)$ does not depend on $\pi_S$ even for moderate sizes $n_S$. Therefore, the lower bound is not as useful as the upper bound.

Finally, if we refine $\epsilon_T$ as we have done for $\sigma_T^2$ in (8), we obtain the following corollary:

**Corollary 8.** *Let $\bar{\epsilon}_T(\rho, \sigma_\mathrm{n}^2, \sigma_\mathrm{n}^2, X_T, X_S) \stackrel{\text{def}}{=} \epsilon_T(1, \sigma_\mathrm{n}^2, \bar{\bar{\sigma}}_\mathrm{n}^2, X_T, X_S)$. Then*
$$\bar{\epsilon}_T(\rho, \sigma_\mathrm{n}^2, \sigma_\mathrm{n}^2, X_T, X_S) \geqslant \epsilon_T(\rho, \sigma_\mathrm{n}^2, \sigma_\mathrm{n}^2, X_T, X_S).$$

## 3.4 Exact computation of generalization error

The factorization of $\sigma_T^2$ expressed by (12) allows the generalization error to be computed exactly in certain cases. We replace the quadratic form in (12) by matrix trace and then integrate out $\boldsymbol{x}_*$ to give

$$\epsilon_T(\rho, \sigma_\mathrm{n}^2, X_T, X_S) = \langle k_{**} \rangle - \mathrm{tr}\left(\Sigma^{-1}\langle \boldsymbol{k}_*^\mathrm{x}(\boldsymbol{k}_*^\mathrm{x})^\mathrm{T}\rangle\right) = \sum_{i=1}^\infty \kappa_i - \mathrm{tr}\left(\Sigma^{-1}M\right),$$

where $\Sigma$ denotes $\Sigma(\rho, \sigma_\mathrm{n}^2, \sigma_\mathrm{n}^2)$, the expectations are taken over $\boldsymbol{x}_*$, and $M$ is an $n$-by-$n$ matrix with

$$M_{pq} \stackrel{\text{def}}{=} \int k^\mathrm{x}(\boldsymbol{x}_p, \boldsymbol{x}_*)\, k^\mathrm{x}(\boldsymbol{x}_q, \boldsymbol{x}_*)\, p(\boldsymbol{x}_*)\mathrm{d}\boldsymbol{x}_* = \sum_{i=1}^\infty \kappa_i^2 \phi_i(\boldsymbol{x}_p)\phi_i(\boldsymbol{x}_q), \quad \text{where } \boldsymbol{x}_p, \boldsymbol{x}_q \in X.$$

When the eigenfunctions $\phi_i(\cdot)$s are not bounded, the infinite-summation expression for $M_{pq}$ is often difficult to use. Nevertheless, analytical results for $M_{pq}$ are still possible in some cases using the integral expression. An example is the case of the squared exponential covariance function with normally distributed $\boldsymbol{x}$, when the integrand is a product of three Gaussians.

## 4 Optimal error for the degenerate case of no training data for primary task

If training examples are provided only for task $S$, then task $T$ has the following optimal performance.

**Proposition 9.** *Under optimal sampling on a 1-d space, if the covariance function satisfies Sacks-Ylvisaker conditions of order $r$, then $\epsilon_T^{opt}(\rho, \sigma^2, 1, n) = \Theta(n_S^{-(2r+1)/(2r+2)}) + (1-\rho^2)\sum_{i=1}^\infty \kappa_i$.*

*Proof.* We obtain $\epsilon_T^{\mathrm{opt}}(\rho, \sigma^2, 1, n) = \rho^2 \epsilon_T^{\mathrm{opt}}(1, \sigma_\mathrm{n}^2, 1, n) + (1-\rho^2)\sum_{i=1}^\infty \kappa_i$ by minimizing Corollary 4 wrt $X_S$. Under the same conditions as the proposition, the optimal generalization error using the single-task GP decays with training set size $n$ as $\Theta(n^{-(2r+1)/(2r+2)})$ [11, Proposition V.3]. Thus $\rho^2 \epsilon_T^{\mathrm{opt}}(1, \sigma_\mathrm{n}^2, 1, n) = \rho^2 \Theta(n_S^{-(2r+1)/(2r+2)}) = \Theta(n_S^{-(2r+1)/(2r+2)})$. □

A directly corollary of the above result is that one cannot expect to do better than $(1-\rho^2)\sum \kappa_i$ on the average. As this is a lower bound, the same can be said for incorrectly specified GP priors.

## 5 Theoretical bounds on learning curve

Using the results from section 3, lower and upper bounds on the learning curve may be computed by averaging over the choice of $X$ using Monte Carlo approximation.[1] For example, using Corollary 2 and integrating wrt $p(X)\mathrm{d}X$ gives the following trivial bounds on the learning curve:

**Corollary 10.** $\epsilon_T^{avg}(1, \sigma_\mathrm{n}^2, \pi_S, n) \leqslant \epsilon_T^{avg}(\rho, \sigma_\mathrm{n}^2, \pi_S, n) \leqslant \epsilon_T^{avg}(0, \sigma_\mathrm{n}^2, \pi_S, n)$.

The gap between the trivial bounds can be analyzed as follows. Recall that $\pi_S n \in \mathbb{N}_0$ by definition, so that $\epsilon_T^{\mathrm{avg}}(1, \sigma_\mathrm{n}^2, \pi_S, (1-\pi_S)n) = \epsilon_T^{\mathrm{avg}}(0, \sigma_\mathrm{n}^2, \pi_S, n)$. Therefore $\epsilon_T^{\mathrm{avg}}(1, \sigma_\mathrm{n}^2, \pi_S, n)$ is equivalent to $\epsilon_T^{\mathrm{avg}}(0, \sigma_\mathrm{n}^2, \pi_S, n)$ scaled along the $n$-axis by the factor $(1-\pi_S) \in [0, 1]$, and hence the gap between the trivial bounds becomes wider with $\pi_S$.

In the rest of this section, we derive non-trivial theoretical bounds on the learning curve before providing simulation results. Theoretical bounds are particularly attractive for high-dimensional input-spaces, on which Monte Carlo approximation is harder.

### 5.1 Lower bound

For the single-task GP, a lower bound on its learning curve is $\sigma_\mathrm{n}^2 \sum_{i=1}^\infty \kappa_i/(\sigma_\mathrm{n}^2 + n\kappa_i)$ [15]. We shall call this the single-task OV bound. This lower bound can be combined with Corollary 6.

**Proposition 11.** $\epsilon_T^{avg}(\rho, \sigma_\mathrm{n}^2, \pi_S, n) \geqslant \rho^2 \sigma_\mathrm{n}^2 \sum_{i=1}^\infty \dfrac{\kappa_i}{\sigma_\mathrm{n}^2 + n\kappa_i} + (1-\rho^2)\sigma_\mathrm{n}^2 \sum_{i=1}^\infty \dfrac{\kappa_i}{\sigma_\mathrm{n}^2 + (1-\pi_S)n\kappa_i}$,

*or equivalently,* $\epsilon_T^{avg}(\rho, \sigma_\mathrm{n}^2, \pi_S, n) \geqslant \sigma_\mathrm{n}^2 \sum_{i=1}^\infty \dfrac{b_i^1 \kappa_i}{\sigma_\mathrm{n}^2 + n\kappa_i}$, *with* $b_i^1 \stackrel{\text{def}}{=} \dfrac{\sigma_\mathrm{n}^2 + (1-\rho^2\pi_S)n\kappa_i}{\sigma_\mathrm{n}^2 + (1-\pi_S)n\kappa_i}$,

*or equivalently,* $\epsilon_T^{avg}(\rho, \sigma_\mathrm{n}^2, \pi_S, n) \geqslant \sigma_\mathrm{n}^2 \sum_{i=1}^\infty \dfrac{b_i^0 \kappa_i}{\sigma_\mathrm{n}^2 + (1-\pi_S)n\kappa_i}$, *with* $b_i^0 \stackrel{\text{def}}{=} \dfrac{\sigma_\mathrm{n}^2 + (1-\rho^2\pi_S)n\kappa_i}{\sigma_\mathrm{n}^2 + n\kappa_i}$.

*Proof sketch.* To obtain the first inequality, we integrate Corollary 6 wrt to $p(X)\mathrm{d}X$, and apply the single-task OV bound twice. For the second inequality, its $i$th summand is obtained by combining the corresponding pair of $i$th summands in the first inequality. The third inequality is obtained from the second by swapping the denominator of $b_i^1$ with that of $\kappa_i/(\sigma_\mathrm{n}^2 + n\kappa_i)$ for every $i$. □

For fixed $\sigma_\mathrm{n}^2$, $\pi_S$ and $n$, denote the above bound by $\mathrm{OV}_\rho$. Then $\mathrm{OV}_0$ and $\mathrm{OV}_1$ are both single task bounds. In particular, from Corollary 10, we have that the $\mathrm{OV}_1$ is a lower bound on $\epsilon_T^\mathrm{avg}(\rho, \sigma_\mathrm{n}^2, \pi_S, n)$. From the first expression of the above proposition, it is clear from the "mixture" nature of the bound that the two-tasks bound $\mathrm{OV}_\rho$ is always better than $\mathrm{OV}_1$. As $\rho^2$ decreases, the two-tasks bound moves towards the $\mathrm{OV}_0$; and as $\pi_S$ increases, the gap between $\mathrm{OV}_0$ and $\mathrm{OV}_1$ increases. In addition, the gap is also larger for rougher processes, which are harder to learn. Therefore, the relative tightness of $\mathrm{OV}_\rho$ over $\mathrm{OV}_1$ is more noticeable for lower $\rho^2$, higher $\pi_S$ and rougher processes.

The second expression in the Proposition 11 is useful for comparing with the $\mathrm{OV}_1$. Each summand for the two-tasks case is a factor $b_i^1$ of the corresponding summand for the single-task case. Since $b_i^1 \in [1, (1 - \rho^2\pi_S)/(1 - \pi_S)[$ , $\mathrm{OV}_\rho$ is more than $\mathrm{OV}_1$ by at most $(1 - \rho^2)\pi_S/(1 - \pi_S)$ times. Similarly, the third expression of the proposition is useful for comparing with $\mathrm{OV}_0$: each summand for the the two-tasks case is a factor $b_i^0 \in ](1 - \rho^2\pi_S), 1]$ of the corresponding single-task one. Hence, $\mathrm{OV}_\rho$ is less than $\mathrm{OV}_0$ by up to $\rho^2\pi_S$ times. In terms of the lower bound, this is the limit to which multi-task learning can outperform the single-task learning that ignores the secondary task.

## 5.2 Upper bound using equivalent noise

An upper bound on the learning curve of a single-task GP is given in [16]. We shall refer to this as the single-task FWO bound and combine it with the approach in section 3.3 to obtain an upper on the learning curve of task $T$. Although the single-task FWO bound was derived for observations with isotropic noise, with some modifications (see supplementary material §S.4), the derivations are still valid for observations with heteroscedastic and correlated noise. Below is a version of the FWO bound that has yet to assume isotropic noise:

**Theorem 12.** *([16], modified second part of Theorem 6) Consider a zero-mean GP with covariance function $k^\mathrm{x}(\cdot, \cdot)$, and eigenvalues $\kappa_i$ and eigenfunctions $\phi_i(\cdot)$ under the measure $p(\boldsymbol{x})\mathrm{d}\boldsymbol{x}$; and suppose that the noise (co)variances of the observations are given by $\gamma^2(\cdot, \cdot)$. For $n$ observations $\{\boldsymbol{x}_i\}_{i=1}^n$, let $H$ and $\Phi$ be matrices such that $H_{ij} \overset{\text{def}}{=} k^\mathrm{x}(\boldsymbol{x}_i, \boldsymbol{x}_j) + \gamma^2(\boldsymbol{x}_i, \boldsymbol{x}_j)$ and $\Phi_{ij} \overset{\text{def}}{=} \phi_j(\boldsymbol{x}_i)$. Then the learning curve at $n$ is upper-bounded by $\sum_{i=1}^\infty \kappa_i - n\sum_{i=1}^\infty \kappa_i^2/c_i$, where $c_i \overset{\text{def}}{=} \langle (\Phi^\mathrm{T} H \Phi)_{ii} \rangle /n$, and the expectation in $c_i$ is taken over the set of $n$ input locations drawn independently from $p(\boldsymbol{x})$.*

Unlike [16], we do not assume that the noise variance $\gamma^2(\boldsymbol{x}_i, \boldsymbol{x}_j)$ is of the form $\sigma_\mathrm{n}^2\delta_{ij}$. Instead of proceeding from the upper bound $\sigma_T^2(1, \sigma_\mathrm{n}^2, \bar{\bar{\sigma}}_\mathrm{n}^2)$, we proceed directly from the exact posterior variance given by (12). Thus we set the observation noise (co)variance $\gamma^2(\boldsymbol{x}_i, \boldsymbol{x}_j)$ to

$$\delta(\boldsymbol{x}_i \in X_T)\delta(\boldsymbol{x}_j \in X_T)\,\delta_{ij}\sigma_\mathrm{n}^2 \;+\; \delta(\boldsymbol{x}_i \in X_S)\delta(\boldsymbol{x}_j \in X_S)\,\left[\beta k^\mathrm{x}(\boldsymbol{x}_i, \boldsymbol{x}_j) + \rho^{-2}\delta_{ij}\sigma_\mathrm{n}^2\right], \quad (14)$$

so that, through the definition of $c_i$ in Theorem 12, we obtain

$$c_i = (1 + \beta\pi_S)\left\{\left[(1 + \beta\pi_S^2)n/(1 + \beta\pi_S) - 1\right]\kappa_i + \int k^\mathrm{x}(\boldsymbol{x}, \boldsymbol{x})\left[\phi_i(\boldsymbol{x})\right]^2 p(\boldsymbol{x})\mathrm{d}\boldsymbol{x} + \sigma_\mathrm{n}^2\right\}; \quad (15)$$

details are in the supplementary material §S.5. This leads to the following proposition:

**Proposition 13.** *Let $\beta \overset{\text{def}}{=} \rho^{-2} - 1$. Then, using the $c_i$s defined in (15), we have*
$$\epsilon_T^\mathrm{avg}(\rho, \sigma_\mathrm{n}^2, \pi_S, n) \leqslant \sum_{i=1}^\infty \kappa_i - n\sum_{i=1}^\infty \kappa_i^2/c_i.$$

Denote the above upper bound by $\mathrm{FWO}_\rho$. When $\rho = \pm 1$ or $\pi_S = 0$, the single-task FWO upper bound is recovered. However, $\mathrm{FWO}_\rho$ with $\rho = 0$ gives the prior variance $\sum \kappa_i$ instead. A trivial upper bound can be obtained using Corollary 10, by replacing $n$ with $(1 - \pi_S)n$ in the single-task FWO bound. The $\mathrm{FWO}_\rho$ bound is better than this trivial single-task bound for small $n$ and high $|\rho|$.

## 5.3 Comparing bounds by simulations of learning curve

We compare our bounds with simulated learning curves. We follow the third scenario in [13]: the input space is one dimensional with Gaussian distribution $\mathcal{N}(0, 1/12)$, the covariance function is the

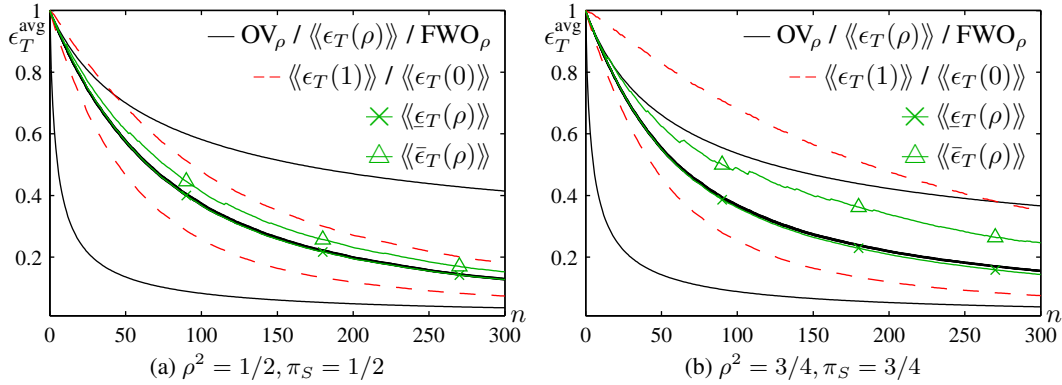

(a) $\rho^2 = 1/2, \pi_S = 1/2$       (b) $\rho^2 = 3/4, \pi_S = 3/4$

Figure 3: Comparison of various bounds for two settings of $(\rho, \pi_S)$. Each graph plots $\epsilon_T^{\text{avg}}$ against $n$ and consists of the "true" multi-task learning curve (middle —), the theoretical lower/upper bounds of Propositions 11/13 (lower/upper —), the empirical trivial lower/upper bounds using Corollary 10 (lower/upper --), and the empirical lower/upper bounds using Corollaries 6/8 ($\times$/$\triangle$). The thickness of the "true" multi-task learning curve reflects 95% confidence interval.

unit variance squared exponential $k^x(x, x') = \exp[-(x - x')^2/(2l^2)]$ with length-scale $l = 0.01$, the observation noise variance is $\sigma_n^2 = 0.05$, and the learning curves are computed for up to $n = 300$ training data points. When required, the average over $\boldsymbol{x}_*$ is computed analytically (see section 3.4). The empirical average over $X \overset{\text{def}}{=} X_T \cup X_S$, denoted by $\langle\!\langle \cdot \rangle\!\rangle$, is computed over 100 randomly sampled training sets. The process eigenvalues $\kappa_i$s needed to compute the theoretical bounds are given in [17]. Supplementary material §S.6 gives further details.

Learning curves for pairwise combinations of $\rho^2 \in \{1/8, 1/4, 1/2, 3/4\}$ and $\pi_S \in \{1/4, 1/2, 3/4\}$ are computed. We compare the following: (a) the "true" multi-task learning curve $\langle\!\langle \epsilon_T(\rho) \rangle\!\rangle$ obtained by averaging $\sigma_T^2(\rho)$ over $\boldsymbol{x}_*$ and $X$; (b) the theoretical bounds $\text{OV}_\rho$ and $\text{FWO}_\rho$ of Propositions 11 and 13; (c) the trivial upper and lower bounds that are single-task learning curves $\langle\!\langle \epsilon_T(0) \rangle\!\rangle$ and $\langle\!\langle \epsilon_T(1) \rangle\!\rangle$ obtained by averaging $\sigma_T^2(0)$ and $\sigma_T^2(1)$; and (d) the empirical lower bound $\langle\!\langle \underline{\epsilon}_T(\rho) \rangle\!\rangle$ and upper bound $\langle\!\langle \bar{\epsilon}_T(\rho) \rangle\!\rangle$ using Corollaries 6 and 8. Figure 3 gives some indicative plots of the curves.

We summarize with the following observations: (a) The gap between the trivial bounds $\langle\!\langle \epsilon_T(0) \rangle\!\rangle$ and $\langle\!\langle \epsilon_T(1) \rangle\!\rangle$ increases with $\pi_S$, as described at the start of section 5. (b) We find the lower bound $\langle\!\langle \underline{\epsilon}_T(\rho) \rangle\!\rangle$ a rather close approximation to the multi-task learning curve $\langle\!\langle \epsilon_T(\rho) \rangle\!\rangle$, as evidenced by the much overlap between the $\times$ lines and the middle — lines in Figure 3. (c) The curve for the empirical upper bound $\langle\!\langle \bar{\epsilon}_T(\rho) \rangle\!\rangle$ using the equivalent noise method has jumps, e.g., the $\triangle$ lines in Figure 3, because the equivalent noise variance $\bar{\bar{\sigma}}_n^2$ increases whenever a datum for $X_S$ is sampled. (d) For small $n$, $\langle\!\langle \epsilon_T(\rho) \rangle\!\rangle$ is closer to $\text{FWO}_\rho$, but becomes closer to $\text{OV}_\rho$ as $n$ increases, as shown by the unmarked solid lines in Figure 3. This is because the theoretical lower bound $\text{OV}_\rho$ is based on the asymptotically exact single-task OV bound and the $\underline{\epsilon}_T(\rho)$ bound, which is observed to approximate the multi-task learning curve rather closely (point (b)).

**Conclusions**    We have measured the influence of the secondary task on the primary task using the generalization error and the learning curve, parameterizing these with the correlation $\rho$ between the two tasks, and the proportion $\pi_S$ of observations for the secondary task. We have provided bounds on the generalization error and learning curves, and these bounds highlight the effects of $\rho$ and $\pi_S$. This is a step towards understanding the role of the matrix $K^f$ of inter-task similarities in multi-task GPs with more than two tasks. Analysis on the degenerate case of no training data for the primary task has uncovered an intrinsic limitation of multi-task GP. Our work contributes to an understanding of multi-task learning that is orthogonal to the existing PAC-based results in the literature.

**Acknowledgments**

I thank E Bonilla for motivating this problem, CKI Williams for helpful discussions and for proposing the equivalent isotropic noise approach, and DSO National Laboratories, Singapore, for financial support. This work is supported in part by the EU through the PASCAL2 Network of Excellence.

**References**

[1] Carl E. Rasmussen and Christopher K. I. Williams. *Gaussian Processes for Machine Learning*. MIT Press, Cambridge, Massachusetts, 2006.

[2] Yee Whye Teh, Matthias Seeger, and Michael I. Jordan. Semiparametric latent factor models. In Robert G. Cowell and Zoubin Ghahramani, editors, *Proceedings of the 10th International Workshop on Artificial Intelligence and Statistics*, pages 333–340. Society for Artificial Intelligence and Statistics, January 2005.

[3] Edwin V. Bonilla, Felix V. Agakov, and Christopher K. I. Williams. Kernel Multi-task Learning using Task-specific Features. In Marina Meila and Xiaotong Shen, editors, *Proceedings of the 11th International Conference on Artificial Intelligence and Statistics*. Omni Press, March 2007.

[4] Kai Yu, Wei Chu, Shipeng Yu, Volker Tresp, and Zhao Xu. Stochastic Relational Models for Discriminative Link Prediction. In B. Schölkopf, J. Platt, and T. Hofmann, editors, *Advances in Neural Information Processing Systems 19*, Cambridge, MA, 2007. MIT Press.

[5] Edwin V. Bonilla, Kian Ming A. Chai, and Christopher K.I. Williams. Multi-task Gaussian process prediction. In J.C. Platt, D. Koller, Y. Singer, and S. Roweis, editors, *Advances in Neural Information Processing Systems 20*. MIT Press, Cambridge, MA, 2008.

[6] Jonathan Baxter. A Model of Inductive Bias Learning. *Journal of Artificial Intelligence Research*, 12:149–198, March 2000.

[7] Andreas Maurer. Bounds for linear multi-task learning. *Journal of Machine Learning Research*, 7:117–139, January 2006.

[8] Shai Ben-David and Reba Schuller Borbely. A notion of task relatedness yielding provable multiple-task learning guarantees. *Machine Learning*, 73(3):273–287, 2008.

[9] Christopher K. I. Williams and Francesco Vivarelli. Upper and lower bounds on the learning curve for Gaussian processes. *Machine Learning*, 40(1):77–102, 2000.

[10] Ya Xue, Xuejun Liao, Lawrence Carin, and Balaji Krishnapuram. Multi-task learning for classification with Dirichlet process prior. *Journal of Machine Learning Research*, 8:35–63, January 2007.

[11] Klaus Ritter. *Average-Case Analysis of Numerical Problems*, volume 1733 of *Lecture Notes in Mathematics*. Springer, 2000.

[12] Christopher T. H. Baker. *The Numerical Treatment of Integral Equations*. Clarendon Press, 1977.

[13] Peter Sollich and Anason Halees. Learning curves for Gaussian process regression: Approximations and bounds. *Neural Computation*, 14(6):1393–1428, 2002.

[14] Noel A. Cressie. *Statistics for Spatial Data*. Wiley, New York, 1993.

[15] Manfred Opper and Francesco Vivarelli. General bounds on Bayes errors for regression with Gaussian processes. In Kearns et al. [18], pages 302–308.

[16] Giancarlo Ferrari Trecate, Christopher K. I. Williams, and Manfred Opper. Finite-dimensional approximation of Gaussian processes. In Kearns et al. [18], pages 218–224.

[17] Huaiyu Zhu, Christopher K. I. Williams, Richard Rohwer, and Michal Morciniec. Gaussian regression and optimal finite dimensional linear models. In Christopher M. Bishop, editor, *Neural Networks and Machine Learning*, volume 168 of *NATO ASI Series F: Computer and Systems Sciences*, pages 167–184. Springer-Verlag, Berlin, 1998.

[18] Michael J. Kearns, Sara A. Solla, and David A. Cohn, editors. *Advances in Neural Information Processing Systems 11*, 1999. The MIT Press.

## Footnotes

[1]Approximate lower bounds are also possible, by combining Corollary 6 and approximations in, e.g., [13].
